# Ensemble Clustering using Semidefinite Programming

**Vikas Singh**
Biostatistics and Medical Informatics
University of Wisconsin – Madison
vsingh @ biostat.wisc.edu

**Lopamudra Mukherjee**
Computer Science and Engineering
State University of New York at Buffalo
lm37 @ cse.buffalo.edu

**Jiming Peng**
Industrial and Enterprise System Engineering
University of Illinois at Urbana-Champaign
pengj @ uiuc.edu

**Jinhui Xu**
Computer Science and Engineering
State University of New York at Buffalo
jinhui @ cse.buffalo.edu

## Abstract

We consider the ensemble clustering problem where the task is to 'aggregate' multiple clustering solutions into a single consolidated clustering that maximizes the shared information among given clustering solutions. We obtain several new results for this problem. First, we note that the notion of agreement under such circumstances can be better captured using an agreement measure based on a $2D$ string encoding rather than voting strategy based methods proposed in literature. Using this generalization, we first derive a nonlinear optimization model to maximize the new agreement measure. We then show that our optimization problem can be transformed into a strict 0-1 Semidefinite Program (SDP) via novel convexification techniques which can subsequently be relaxed to a polynomial time solvable SDP. Our experiments indicate improvements not only in terms of the proposed agreement measure but also the existing agreement measures based on voting strategies. We discuss evaluations on clustering and image segmentation databases.

## 1 Introduction

In the so-called Ensemble Clustering problem, the target is to 'combine' multiple clustering solutions or partitions of a set into a single consolidated clustering that maximizes the information shared (or 'agreement') among all available clustering solutions. The need for this form of clustering arises in many applications, especially real world scenarios with a high degree of uncertainty such as image segmentation with poor signal to noise ratio and computer assisted disease diagnosis. It is quite common that a single clustering algorithm may not yield satisfactory results, while multiple algorithms may individually make imperfect choices, assigning some elements to wrong clusters. Usually, by considering the results of several *different* clustering algorithms *together*, one may be able to mitigate degeneracies in individual solutions and consequently obtain better solutions. The idea has been employed successfully for microarray data classification analysis [1], computer assisted diagnosis of diseases [2] and in a number of other applications [3].

Formally, given a data set $D = \{d_1, d_2, \ldots, d_n\}$, a set of clustering solutions $C = \{C_1, C_2, \ldots, C_m\}$ obtained from $m$ different clustering algorithms is called a *cluster ensemble*. Each solution, $C_i$, is the partition of the data into at most $k$ different clusters. The *Ensemble Clustering* problem requires one to use the individual solutions in $C$ to partition $D$ into $k$ clusters such that information shared ('agreement') among the solutions of $C$ is maximized.

## 1.1 Previous works

The Ensemble Clustering problem was recently introduced by Strehl and Ghosh [3], in [4] a related notion of correlation clustering was independently proposed by Bansal, Blum, and Chawla. The problem has attracted a fair amount of attention and a number of interesting techniques have been proposed [3, 2, 5, 6], also see [7, 4]. Formulations primarily differ in how the objective of *shared information maximization* or *agreement* is chosen, we review some of the popular techniques next.

The *Instance Based Graph Formulation* (IBGF) [2, 5] first constructs a fully connected graph $G = (V, W)$ for the ensemble $C = (C_1, \ldots, C_m)$, each node represents an element of $D = \{d_1, \ldots, d_n\}$. The edge weight $w_{ij}$ for the pair $(d_i, d_j)$ is defined as the number of algorithms in $C$ that assign the nodes $d_i$ and $d_j$ to the same cluster (i.e., $w_{ij}$ measures the *togetherness* frequency of $d_i$ and $d_j$). Then, standard graph partitioning techniques are used to obtain a final clustering solution. In *Cluster Based Graph Formulation* (CBGF), a given cluster ensemble is represented as $C = \{C_{11}, \ldots, C_{mk}\} = \{\bar{C}_1, \ldots, \bar{C}_{m \cdot k}\}$ where $C_{ij}$ denotes the $i$th cluster of the $j$th algorithm in $C$. Like IBGF, this approach also constructs a graph, $G = (V, W)$, to model the correspondence (or 'similarity') relationship among the $mk$ clusters, where the similarity matrix $W$ reflects the Jaccard's similarity measure between clusters $\bar{C}_i$ and $\bar{C}_j$. The graph is then partitioned so that the clusters of the same group are similar to one another. Variants of the problem have also received considerable attention in the theoretical computer science and machine learning communities. A recent paper by Ailon, Charikar, and Newman [7] demonstrated connections to other well known problems such as Rank Aggregation, their algorithm is simple and obtains an expected constant approximation guarantee (via linear programming duality). In addition to [7], other results include [4, 8].

A commonality of existing algorithms for Ensemble Clustering [3, 2, 9] is that they employ a graph construction, as a first step. Element pairs (cluster pairs or item pairs) are then evaluated and their edges are assigned a weight that reflects their similarity. A natural question relates to whether we can find a better representation of the available information. This will be the focus of the next section.

## 2 Key Observations: Two is a company, is three a crowd?

Consider an example where one is 'aggregating' recommendations made by a group of family and friends for dinner table seating assignments at a wedding. The hosts would like each 'table' to be able to find a common topic of dinner conversation. Now, consider three persons, Tom, Dick, and Harry invited to this reception. Tom and Dick share a common interest in Shakespeare, Dick and Harry are both surfboard enthusiasts, and Harry and Tom attended college together. Because they had strong pairwise similarities, they were seated together but had a rather dull evening.

A simple analysis shows that the three guests had strong common interests when considered two at a time, but there was weak communion as a group. The connection of this example to the ensemble clustering problem is clear. Existing algorithms represent the similarity between elements in $D$ as a scalar value assigned to the edge joining their corresponding nodes in the graph. This weight is essentially a 'vote' reflecting the number of algorithms that assigned those two elements to the same cluster. The mechanism seems perfect until we ask if strong pairwise coupling necessarily implies coupling for a larger group as well. The weight metric considering two elements does not retain information about which algorithms assigned them together. When expanding the group to include more elements, one is not sure if a common feature exists under which the larger group is similar. It seems natural to assign a higher priority to triples or larger groups of people that were recommended to be seated together (must be similar under at least one feature) compared to groups that were never assigned to the same table by any person in the recommendation group (clustering algorithm), notwithstanding pairwise evaluations, for an illustrative example see [10]. While this problem seems to be a distinctive disadvantage for only the IBGF approach; it also affects the CBGF approach. This can be seen by looking at clusters as items and the Jaccard's similarity measure as the vote (weight) on the edges.

## 3 Main Ideas

To model the intuition above, we generalize the similarity metric to maximize similarity or 'agreement' by an appropriate encoding of the solutions obtained from individual clustering algorithms.

More precisely, in our generalization the similarity is no longer just a scalar value but a $2D$ string. The ensemble clustering problem thus reduces to a form of string clustering problem where our objective is to assign *similar* strings to the same cluster.

The encoding into a string is done as follows. The data item set is given as $D$ with $|D| = n$. Let $m$ be the number of clustering algorithms with each solution having no more than $k$ clusters. We represent all input information (ensemble) as a single $3D$ matrix, $A \in \Re^{n \times m \times k}$. For every data element $d_l \in D$, $A_l \in \Re^{m \times k}$ is a matrix whose elements are defined by

$$A_l(i, j) = \begin{cases} 1 & \text{if } d_l \text{ is assigned to cluster } i \text{ by } C_j; \\ 0 & \text{otherwise} \end{cases} \tag{1}$$

It is easy to see that the summation of every row of $A_l$ equals 1. We call each $A_l$ an $A$-string. Our goal is to cluster the elements $D = \{d_1, d_2, \ldots, d_n\}$ based on the similarity of their $A$-strings.

We now consider how to compute the clusters based on the similarity (or dissimilarity) of strings. We note that the paper [11] by Gasieniec et al., discussed the so-called Hamming radius $p$-clustering and Hamming diameter $p$-clustering problems on strings. Though their results shed considerable light on the hardness of string clustering with the selected distance measures, those techniques cannot be directly applied to the problem at hand because the objective here is fairly different from the one in [11]. Fortunately, our analysis reveals that a simpler objective is sufficient to capture the essence of similarity maximization in clusters using certain special properties of the $A$-strings.

Our approach is partly inspired by the classical $k$-means clustering where all data points are assigned to the cluster based on the shortest distance to the cluster center. Imagine an ideal input instance for the ensemble clustering problem (all clustering algorithms behave similarly) – one with only $k$ unique members among $n$ $A$-strings. The partitioning simply assigns similar strings to the same partition. The representative for each cluster will then be exactly like its members, is a valid $A$-string, and can be viewed as a *center* in a geometric sense. General input instances will obviously be non-ideal and are likely to contain far more than $k$ unique members. Naturally, the centers of the clusters will vary from its members. This variation can be thought of as *noise* or *disagreement* within the clusters, our objective is to find a set of clusters (and centers) such that the noise is minimized and we move very close to the ideal case. To model this, we consider the centers to be in the *same high dimensional space* as the $A$-strings in $D$ (though it may not belong to $D$). Consider an example where a cluster $i$ in this optimal solution contains items $(d_1, d_2, \ldots, d_7)$. A certain algorithm $C_j$ in the input ensemble clusters items $(d_1, d_2, d_3, d_4)$ in cluster $s$ and $(d_5, d_6, d_7)$ in cluster $p$. How would $C_j$ behave if evaluating the center of cluster $i$ as a data item? The probability it assigns the center to cluster $s$ is $4/7$ and the probability it assigns the center to cluster $p$ is $3/7$. If we emulate this logic – we must pick the choice with the higher probability and assign the center to such a cluster. It can be verified that this choice minimizes the dissent of all items in cluster $i$ to the center. The $A$-string for the center of cluster $i$ will have a "1" at position $(j, s)$. The assignment of $A$-string (items) to clusters is unknown; however, if it were somehow known, we could find the centers for all other clusters $i \in [1, k]$ by computing the average value at every cell of the $A$ matrices corresponding to the members of the cluster and rounding the largest value in every row to 1 (rest to 0) and assigning this as the cluster center. Hence, the dissent within a cluster can be quantified simply by averaging the matrices of elements that belong to the cluster and computing the difference to the center. Our goal is to find such an assignment and group the $A$-strings so that the sum of the absolute differences of the averages of clusters to their centers (dissent) is minimized. In the subsequent sections, we will introduce our optimization framework for ensemble clustering based on these ideas.

## 4  Integer Program for Model 1

We start with a discussion of an Integer Program (IP, for short) formulation for ensemble clustering. For convenience, we denote the final clustering solution by $C^* = \{C_1^*, \ldots, C_k^*\}$ and $C_{ij}$ denotes the cluster $i$ by the algorithm $j$. The variables that constitute the IP are as follows.

$$X_{li'} = \begin{cases} 1 & \text{if } d_l \in C_{i'}^*; \\ 0 & \text{otherwise} \end{cases} \tag{2}$$

$$s_{iji'} = \begin{cases} 1 & \text{if } C_{i'}^* = \arg \max_{i=1,\ldots,k} \{|C_{i'}^* \bigcap C_{ij}|\} \\ 0 & \text{otherwise} \end{cases} \tag{3}$$

We mention that the above definition implies that for a fixed index $i'$, its center, $s_{iji'}$ also provides an indicator to the cluster most similar to $C_{i'}^*$ in the set of clusters produced by the clustering algorithm $C_j$. We are now ready to introduce the following IP.

$$\min \quad \sum_{i'=1}^{k}\sum_{i=1}^{k}\sum_{j=1}^{m} \left| s_{iji'} - \frac{\sum_{l=1}^{n} A_{lij} X_{li'}}{\sum_{l=1}^{n} X_{li'}} \right| \tag{4}$$

$$s.t. \quad \sum_{i'=1}^{k} X_{li'} = 1 \quad \forall l \in [1,n], \quad \sum_{l=1}^{n} X_{li'} \geq 1 \quad \forall i' \in [1,k], \tag{5}$$

$$\sum_{i=1}^{k} s_{iji'} = 1 \quad \forall j \in [1,m], i' \in [1,k], \quad X_{li'} \in \{0,1\}, \quad s_{iji'} \in \{0,1\}. \tag{6}$$

(4) minimizes the sum of the difference between $s_{iji'}$ (the center for cluster $C_{i'}^*$) and the average of all $A_{lij}$ bits of the data elements $d_l$ assigned to cluster $C_{i'}^*$. Recall that $s_{iji'}$ will be 1 if $C_{ij}$ is the most similar cluster to $C_{i'}^*$ among all the clusters produced by algorithm $C_j$. Hence, if $s_{iji'} = 0$ and $\frac{\sum_{l=1}^{n} A_{lij} X_{li'}}{\sum_{l=1}^{n} X_{li'}} \neq 0$, the value $\left| s_{iji'} - \frac{\sum_{l=1}^{n} A_{lij} X_{li'}}{\sum_{l=1}^{n} X_{li'}} \right|$ represents the percentage of data elements in $C_{i'}^*$ that *do not* consent with the majority of the other elements in the group w.r.t. the clustering solution provided by $C_j$. In other words, we are trying to minimize the dissent and maximize the consent simultaneously. The remaining constraints are relatively simple – (5) enforces the condition that a data element should belong to precisely one cluster in the final solution and that every cluster must have size at least 1; (6) ensures that $s_{iji'}$ is an appropriate $A$-string for every cluster center.

## 5   0-1 **Semidefinite Program for Model 1**

The formulation given by (4)-(6) is a mixed integer program (MIP, for short) with a nonlinear objective function in (4). Solving this model optimally, however, is extremely challenging – (a) the constraints in (5)-(6) are discrete; (b) the objective is nonlinear *and* nonconvex. One possible way of attacking the problem is to 'relax' it to some polynomially solvable problems such as SDP (the problem of minimizing a linear function over the intersection of a polyhedron and the cone of symmetric and positive semidefinite matrices, see [12] for an introduction). Our effort would be to convert the nonlinear form in (4) into a 0-1 SDP form. By introducing artificial variables, we rewrite (4) as

$$\min \quad \sum_{i=1}^{k}\sum_{j=1}^{m}\sum_{i'=1}^{k} t_{iji'} \tag{7}$$

$$s_{iji'} - c_{iji'} \leq t_{iji'}, \quad c_{iji'} - s_{iji'} \leq t_{iji'} \quad \forall i, i', j, \tag{8}$$

where the term $c_{iji'}$ represents the second term in (4) defined by

$$c_{iji'} = \frac{\sum_{l=1}^{n} A_{lij} X_{li'}}{\sum_{l=1}^{n} X_{li'}} \quad \forall i, i', j. \tag{9}$$

Since both $A_{lij}$ and $X_{li'}$ are binary, (9) can be rewritten as

$$c_{iji'} = \frac{\sum_{l=1}^{n} A_{lij}^2 X_{li'}^2}{\sum_{l=1}^{n} X_{li'}^2} \quad \forall i, i', j. \tag{10}$$

Let us introduce a matrix variable $y_{i'} \in \Re^n$ whose $l$th column is defined by

$$y_{i'}^{(l)} = \frac{X_{li'}}{\sqrt{\sum_{l=1}^{n} X_{li'}^2}} = \frac{X_{li'}}{\|X_{i'}\|_2}. \tag{11}$$

Let $A_{ij} \in \Re^n$ be a vector whose $l$th element has value $A_l(i,j)$. This allows us to represent (10) as

$$c_{iji'} = \operatorname{tr}(B_{ij} Z_{i'}), \quad Z_{i'}^2 = Z_{i'}, \quad Z_{i'} \succeq 0, \tag{12}$$

where $B_{ij} = \operatorname{diag}(A_{ij})$ is a diagonal matrix with $(B_{ij})_{ll} = A_l(i,j)$, the second and third properties follow from $Z_{i'} = y_{i'} y_{i'}^T$ being a positive semidefinite matrix. Now, we rewrite the constraints for $X$ in terms of $Z$. (5) is automatically satisfied by the following constraints on the elements of $Z_{i'}$.

$$\sum_{l=1}^{n} Z_{i'}^{(ll)} = 1 \quad \forall i' \in [1,k], \quad \sum_{l'=1}^{n} Z_{i'}^{(ll')} \leq 1 \quad \forall i' \in [1,k], \forall l \in [1,n]. \tag{13}$$

where $Z_{i'}^{(uv)}$ refers to the $(u, v)$ entry of matrix $Z_{i'}$. Since $Z_i'$ is a symmetric projection matrix by construction, (7)-(13) constitute a precisely defined 0-1 SDP that can be expressed in trace form as

$$\min \quad \sum_{i'=1}^{k} \text{tr}(\text{diag}(T_{i'}e_k)) \tag{14}$$

$$s.t. \quad (S_{i'} - T_{i'} - Q_{i'}) \leq 0, \quad (Q_{i'} - S_{i'} - T_{i'}) \leq 0 \quad \forall i' \in [1, k], \tag{15}$$

$$(\sum_{i'=1}^{k} Z_{i'})e_n = e_n \ \forall i' \in [1, k], \quad \text{tr}(Z_{i'}) = 1 \ \forall i' \in [1, k], \quad \text{tr}(\sum_{i'=1}^{k} Z_{i'}) = k, \tag{16}$$

$$S_{i'}e_k = e_m \ \forall i' \in [1, k], \quad Z \geq 0; \quad Z_{i'}^2 = Z_{i'}; \quad Z_{i'} = Z_{i'}^T; \quad S_{i'} \in \{0, 1\}. \tag{17}$$

where $Q_{i'}(i, j) = c_{iji'} = \text{tr}(B_{ij}Z_{i'})$, and $e_n \in \Re^n$ is a vector of all $1s$.

The experimental results for this model indicate that it performs very well in practice (see [10]). However, because we must solve the model while maintaining the requirement that $S_{i'}$ be binary (otherwise, the problem becomes ill-posed), a branch and bound type method is needed. Such approaches are widely used in many application areas, but its worst case complexity is exponential in the input data size. In the subsequent sections, we will make several changes to this framework based on additional observations in order to obtain a polynomial algorithm for the problem.

# 6 Integer Program and 0-1 Semidefinite Program for Model 2

Recall the definition of the variables $c_{iji'}$, which can be interpreted as the size of the overlap between the cluster $C_{i'}^*$ in the final solution and $C_{ij}$, and is proportional to the cardinality of $C_{i'}^*$. Let us define

$$c_{i^*ji'} = \max_{i=1,\ldots,k} c_{iji'}.$$

Let us also define vector variables $q_{ji'}$ whose $i$th element is $s_{iji'} - c_{iji'}$. In the IP model 1, we try to minimize the sum of all the $L_1$-norms of $q_{ji'}$. The main difficulty in the previous formulation stems from the fact that $c_{iji'}$ is a fractional function w.r.t the assignment matrix $X$. Fortunately, we note that since entries of $c_{iji'}$ are fractional satisfying $\sum_{i=1}^{k} c_{iji'} = 1$ for any fixed $j, i'$, their sum of squares is maximized when its largest entry is as high as possible. Thus, minimizing the function $1 - \sum_{i=1}^{k}(c_{iji'})^2$ is a reasonable substitute to minimizing the sum of the $L_1$-norms in the IP model 1. The primary advantage of this observation is that we do not need to know the 'index' $(i^*)$ of the maximal element $c_{i^*ji'}$. As before, $X$ denotes the assignment matrix. We no longer need the variable $s$, as it can be easily determined from the solution. This yields the following IP.

$$\min \quad \sum_{i'=1}^{k} \sum_{j=1}^{m} (\sum_{l=1}^{n} X_{li'}) \left(1 - \sum_{i=1}^{k}(c_{iji'})^2\right) \tag{18}$$

$$s.t. \quad \sum_{i'=1}^{k} X_{li'} = 1 \quad \forall l \in [1, n], \quad \sum_{l=1}^{n} X_{li'} \geq 1 \quad \forall i' \in [1, k], \quad X_{li'} \in \{0, 1\}. \tag{19}$$

We next discuss how to transform the above problem to a 0-1 SDP. For this, we first note that the objective function (18) can be expressed as follows.

$$\min \sum_{i'=1}^{k} \sum_{j=1}^{m} \left( (\sum_{l=1}^{n} X_{li'}) - \sum_{i=1}^{k} \frac{(\sum_{l=1}^{n} A_{lij}X_{li'})^2}{\sum_{l=1}^{n} X_{li'}} \right), \tag{20}$$

which can be equivalently stated as

$$\min \left( nm - \sum_{i'=1}^{k} \sum_{j=1}^{m} \sum_{i=1}^{k} \frac{(\sum_{l=1}^{n} A_{lij}X_{li'})^2}{\sum_{l=1}^{n} X_{li'}} \right), \tag{21}$$

The numerator of the second term above can be rewritten as

$$(\sum_{l=1}^{n} A_{lij}X_{li'})^2 = (A_{1ij}X_{1i'} + \ldots + A_{nij}X_{ni'})^2 = (A_{ij}^T X_{i'})^2 = X_{i'}^T A_{ij}A_{ij}^T X_{i'}, \tag{22}$$

where $X_i'$ is the $i'$th column vector of $X$. Therefore, the second term of (21) can be written as

$$
\begin{aligned}
&= \operatorname{tr}\Big(\sum_{i'=1}^{k}\sum_{j=1}^{m}\sum_{i=1}^{k} X_{i'}^T A_{ij} A_{ij}^T X_{i'} (X_{i'}^T X_{i'})^{-1}\Big) \\
&= \operatorname{tr}\Big(\sum_{i'=1}^{k}\sum_{j=1}^{m}\sum_{i=1}^{k} A_{ij} A_{ij}^T Z_{i'}\Big) = \operatorname{tr}\Big(\sum_{j=1}^{m}\sum_{i=1}^{k} A_{ij} A_{ij}^T Z\Big) = \operatorname{tr}\Big(\sum_{j=1}^{m} B_j Z\Big) = \operatorname{tr}(BZ). \quad (23)
\end{aligned}
$$

In (23), $Z_{i'} = X_{i'} (X_{i'}^T X_{i'})^{-1} X_{i'}^T$ (same as in IP model 1) and $Z = \sum_{i'=1}^{k} Z_i'$ and $B = \sum_{j=1}^{m} B_j$. Since each matrix $Z_{i'}$ is a symmetric projection matrix and $X_{i_1'}$ and $X_{i_2'}$ are orthogonal to each other when $i_1' \neq i_2'$, $Z$ is a projection matrix of the form $X(X^T X)^{-1} X$. The last fact also used in [13] is originally attributed to an anonymous referee in [14]. Finally, we derive the 0-1 SDP formulation for the problem (18)-(19) as follows.

$$
\begin{aligned}
\min \quad & (nm - \operatorname{tr}(BZ)) & (24) \\
s.t. \quad & Ze_n = e_n \quad \forall i' \in [1, k], & (25) \\
& \operatorname{tr}(Z) = k, \quad Z \geq 0; \quad Z^2 = Z; \quad Z = Z^T. & (26)
\end{aligned}
$$

**Relaxing and Solving the** 0-1 **SDP:** The relaxation to (24)-(26) exploits the fact that $Z$ is a projection matrix satisfying $Z^2 = Z$. This allows replacing the last three constraints in (26) as $I \succeq Z \succeq 0$. By establishing the result that any feasible solution to the second formulation of 0-1 SDP, $Z^{\texttt{feas}}$ is a rank $k$ matrix, we first solve the relaxed SDP using SeDuMi [15], take the rank $k$ projection of $Z^*$ and then adopt a rounding based on a variant of the winner-takes-all approach to obtain a solution in polynomial time. For the technical details and their proofs, please refer to [10].

## 7 Experimental Results

Our experiments included evaluations on several classification datasets, segmentation databases and simulations. Due to space limitations, we provide a brief summary here. Our first set of experiments illustrates an application to several datasets from the UCI Machine Learning Repository: (1) Iris dataset, (2) Soybean dataset and (3) Wine dataset; these include ground truth data, see http://www.ics.uci.edu/ mlearn/MLRepository.html. To create the ensemble, we used a set of $[4, 10]$ clustering schemes (by varying the clustering criterion and/or algorithm) from the CLUTO clustering toolkit. The multiple solutions comprised the input ensemble, our model was then used to determine a agreement maximizing solution. The ground-truth data was used at this stage to evaluate accuracy of the ensemble (and individual schemes). The results are shown in Figure 1(a)-(c). For each case, we can see that the ensemble clustering solution is at least as good as the best clustering algorithm. Observe, however, that while such results are expected for this and many other datasets (see [3]), the consensus solution may not *always* be superior to the 'best' clustering solution. For instance, in Fig. 1(c) (for $m = 7$) the best solution has a marginally lower error rate than the ensemble. An ensemble solution is useful because we do not know a priori that which algorithm will perform the best (especially if ground truth is unavailable).

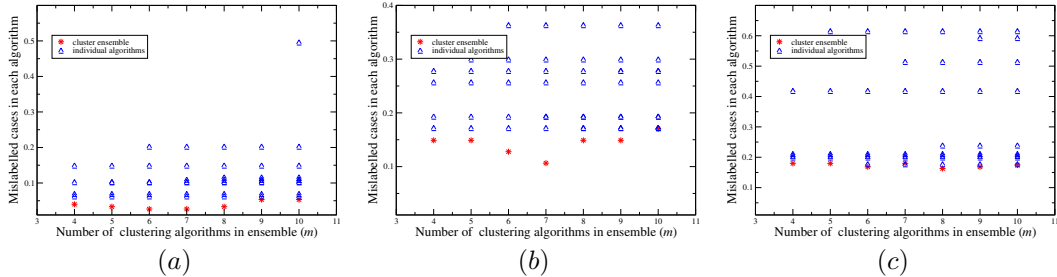

Figure 1: Synergy. The fraction of mislabeled cases ($[0, 1]$) in a consensus solution ($*$) is compared to the number of mislabelled cases ($\Delta$) in individual clustering algorithms. We illustrate the ensemble effect for the Iris dataset in (a), the Soybean dataset in (b), and the Wine dataset in (c).

Our second set of experiments focuses on a novel application of ensembles to the problem of image segmentation. Even sophisticated segmentation algorithms may yield 'different' results on the same image, when multiple segmentations are available, it seems reasonable to 'combine' segmentations to reduce degeneracies. Our experimental results indicate that in many cases, we can obtain a better overall segmentation that captures (more) details in the images more accurately with fewer outlying clusters. In Fig. 2, we illustrate the results on an image from the Berkeley dataset. The segmentations were generated using several powerful algorithms including (a) Normalized Cuts, (b) Energy Minimization by Graph Cuts and (c)–(d) Curve Evolution. Notice that each algorithm performs well but misses out on some details. For instance, (a) and (d) do not segment the eyes; (b) does well in segmenting the shirt collar region but can only recognize one of the eyes and (c) creates an additional cut across the forehead. The ensemble (extreme right) is able to segment these details (eyes, shirt collar and cap) nicely by combining (a)–(d). For implementation details of the algorithm including settings, preprocessing and additional evaluations, please refer to [10].

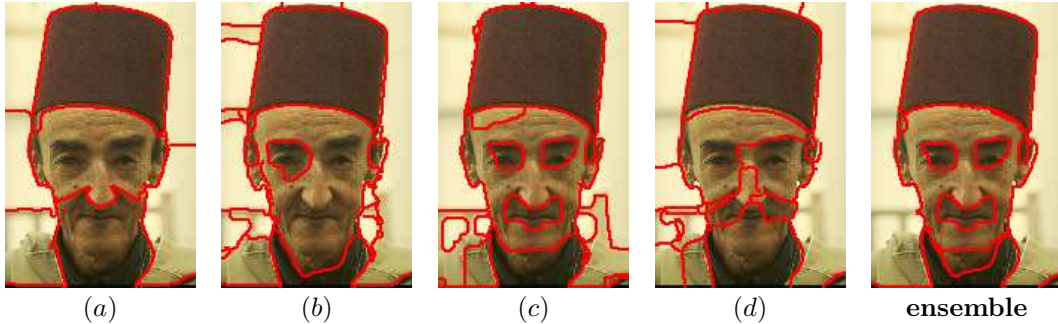

| $(a)$ | $(b)$ | $(c)$ | $(d)$ | **ensemble** |

Figure 2: A segmentation ensemble on an image from the Berkeley Segmentation dataset. (a)–(d) show the individual segmentations overlaid on the input image, the right-most image shows the segmentation generated from ensemble clustering.

The final set of our experiments were performed on $500$ runs of artificially generated cluster ensembles. We first constructed an initial set segmentation, this was then repeatedly permuted (up to $15\%$) yielding a set of clustering solutions. The solutions from our model and [3] were compared w.r.t. our objective functions and Normalized Mutual Information used in [3]. In Figure 3(a), we see that our algorithm (Model 1) outperforms [3] on all instances. In the average case, the ratio is slightly more than $1.5$. We must note the time-quality trade-off because solving Model 1 requires a branch-and-bound approach. In Fig. 3(b), we compare the results of [3] with solutions from the relaxed SDP Model 2 on (24). We can see that our model performs better in $\sim 95\%$ cases. Finally, Figure 1(b) shows a comparison of relaxed SDP Model 2 with [3] on the objective function optimized in [3] (using best among two techniques). We observed that our solutions achieve superior results in $80\%$ of the cases. The results show that even empirically our objective functions model similarity rather well, and that Normalized Mutual Information may be implicitly optimized within this framework.
**Remarks.** We note that the graph partitioning methods used in [3] are typically much faster than the time needed by SDP solvers (e.g., SeDuMi [15] and SDPT3) for comparable problem sizes. However, given the increasing interest in SDP in the last few years, we may expect the development of new algorithms, and faster/more efficient software tools.

## 8  Conclusions

We have proposed a new algorithm for ensemble clustering based on a SDP formulation. Among the important contributions of this paper is, we believe, the observation that the notion of agreement in an ensemble can be captured better using string encoding rather than a voting strategy. While a partition problem defined directly on such strings yields a non-linear optimization problem, we illustrate a transformation into a strict 0-1 SDP via novel convexification techniques. The last result of this paper is the design of a modified model of the SDP based on additional observations on the structure of the underlying matrices. We discuss extensive experimental evaluations on simulations and real datasets, in addition, we illustrate application of the algorithm for segmentation ensembles. We feel that the latter application is of independent research interest; to the best of our knowledge, this is the first algorithmic treatment of generating segmentation ensembles for improving accuracy.

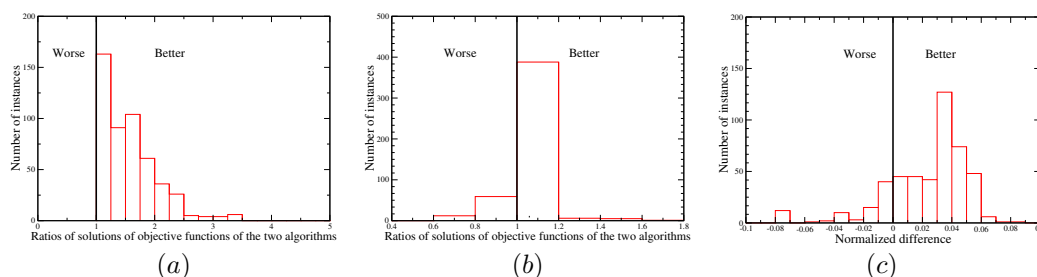

Figure 3: A comparison of [3] with SDP Model 1 in (a), and with SDP Model 2 on (24) in (b). The solution from [3] was used as the numerator. In (c), comparisons (difference in normalized values) between our solution and the best among IBGF and CBGF based on the Normalized Mutual Information (NMI) objective function used in [3].

**Acknowledgments.** This work was supported in part by NSF grants CCF-0546509 and IIS-0713489. The first author was also supported by start-up funds from the Dept. of Biostatistics and Medical Informatics, UW – Madison. We thank D. Sivakumar for useful discussions, Johan Löfberg for a thorough explanation of the salient features of Yalmip [16], and the reviewers for suggestions regarding the presentation of the paper. One of the reviewers also pointed out a typo in the derivations in §6.

# References

[1] V. Filkov and S. Skiena. Integrating microarray data by consensus clustering. In *Proc. of International Conference on Tools with Artificial Intelligence*, page 418, 2003.

[2] X. Z. Fern and C. E. Brodley. Solving cluster ensemble problems by bipartite graph partitioning. In *Proc. of International Conference on Machine Learning*, page 36, 2004.

[3] A. Strehl and J. Ghosh. Cluster Ensembles – A Knowledge Reuse Framework for Combining Partitionings. In *Proc. of AAAI 2002*, pages 93–98, 2002.

[4] N. Bansal, A. Blum, and S. Chawla. Correlation clustering. In *Proc. Symposium on Foundations of Computer Science*, page 238, 2002.

[5] S. Monti, P. Tamayo, J. Mesirov, and T. Golub. Consensus clustering: A resampling-based method for class discovery and visualization of gene expression microarray data. *Mach. Learn.*, 52(1-2):91–118, 2003.

[6] A. Gionis, H. Mannila, and P. Tsaparas. Clustering aggregation. In *Proc. of International Conference on Data Engineering*, pages 341–352, 2005.

[7] N. Ailon, M. Charikar, and A. Newman. Aggregating inconsistent information: ranking and clustering. In *Proc. of Symposium on Theory of Computing*, pages 684–693, 2005.

[8] M. Charikar, V. Guruswami, and A. Wirth. Clustering with qualitative information. *J. Comput. Syst. Sci.*, 71(3):360–383, 2005.

[9] X. Z. Fern and C. E. Brodley. Random projection for high dimensional data clustering: A cluster ensemble approach. In *Proceedings of International Conference on Machine Learning*, 2003.

[10] V. Singh. *On Several Geometric Optimization Problems in Biomedical Computation*. PhD thesis, State University of New York at Buffalo, 2007.

[11] L. Gasieniec, J. Jansson, and A. Lingas. Approximation algorithms for hamming clustering problems. In *Proc. of Symposium on Combinatorial Pattern Matching*, pages 108–118, 2000.

[12] S. Boyd and L. Vandenberghe. *Convex Optimization*. Cambridge University Press, New York, 2004.

[13] J. Peng and Y. Wei. Approximating k-means-type clustering via semidefinite programming. *SIAM Journal on Optimization*, 18(1):186–205, 2007.

[14] A. D. Gordon and J. T. Henderson. An algorithm for euclidean sum of squares classification. *Biometrics*, 33:355–362, 1977.

[15] J. F. Sturm. Using SeDuMi 1.02, A Matlab Toolbox for Optimization over Symmetric Cones. *Optimization Methods and Software*, 11-12:625–653, 1999.

[16] J. Löfberg. YALMIP : A toolbox for modeling and optimization in MATLAB. In *CCA/ISIC/CACSD*, September 2004.